# A Cost-Shaping LP for Bellman Error Minimization with Performance Guarantees

**Daniela Pucci de Farias**
Mechanical Engineering
Massachusetts Institute of Technology

**Benjamin Van Roy**
Management Science and Engineering
and Electrical Engineering
Stanford University

## Abstract

We introduce a new algorithm based on linear programming that approximates the differential value function of an average-cost Markov decision process via a linear combination of pre-selected basis functions. The algorithm carries out a form of cost shaping and minimizes a version of Bellman error. We establish an error bound that scales gracefully with the number of states without imposing the (strong) Lyapunov condition required by its counterpart in [6]. We propose a path-following method that automates selection of important algorithm parameters which represent counterparts to the "state-relevance weights" studied in [6].

## 1 Introduction

Over the past few years, there has been a growing interest in linear programming (LP) approaches to approximate dynamic programming (DP). These approaches offer algorithms for computing weights to fit a linear combination of pre-selected basis functions to a dynamic programming value function. A control policy that is "greedy" with respect to the resulting approximation is then used to make real-time decisions.

Empirically, LP approaches appear to generate effective control policies for high-dimensional dynamic programs [1, 6, 11, 15, 16]. At the same time, the strength and clarity of theoretical results about such algorithms have overtaken counterparts available for alternatives such as approximate value iteration, approximate policy iteration, and temporal-difference methods. As an example, a result in [6] implies that, for a discrete-time finite-state Markov decision process (MDP), if the span of the basis functions contains the constant function and comes within a distance of $\epsilon$ of the dynamic programming value function then the approximation generated by a certain LP will come within a distance of $O(\epsilon)$. Here, the coefficient of the $O(\epsilon)$ term depends on the discount factor and the metric used for measuring distance, but not on the choice of basis functions. On the other hand, the strongest results available for approximate value iteration and approximate policy iteration only promise $O(\epsilon)$ error under additional requirements on iterates generated in the course of executing

the algorithms [3, 13]. In fact, it has been shown that, even when $\epsilon = 0$, approximate value iteration can generate a diverging sequence of approximations [2, 5, 10, 14].

In this paper, we propose a new LP for approximating optimal policies. We work with a formulation involving average cost optimization of a possibly infinite-state MDP. The fact that we work with this more sophisticated formulation is itself a contribution to the literature on LP approaches to approximate DP, which have been studied for the most part in finite-state discounted-cost settings. But we view as our primary contributions the proposed algorithms and theoretical results, which strengthen in important ways previous results on LP approaches and unify certain ideas in the approximate DP literature. In particular, highlights of our contributions include:

1. **Relaxed Lyapunov Function dependence.** Results in [6] suggest that – in order for the LP approach presented there to scale gracefully to large problems – a certain linear combination of the basis functions must be a "Lyapunov function," satisfying a certain strong Lyapunov condition. The method and results in our current paper eliminate this requirement. Further, the error bound is strengthened because it alleviates an undesirable dependence on the Lyapunov function that appears in [6] even when the Lyapunov condition is satisfied.

2. **Restart Distribution Selection.** Applying the LP studied in [6] requires manual selection of a set of parameters called *state-relevance weights*. That paper illustrated the importance of a good choice and provided intuition on how one might go about making the choice. The LP in the current paper does not explicitly make use of state-relevance weights, but rather, an analog which we call a *restart distribution*, and we propose an automated method for finding a desirable restart distribution.

3. **Relation to Bellman-Error Minimization.** An alternative approach for approximate DP aims at minimizing "Bellman error" (this idea was first suggested in [16]). Methods proposed for this (e.g., [4, 12]) involve stochastic steepest descent of a complex nonlinear function. There are no results indicating whether a global minimum will be reached or guaranteeing that a local minimum attained will exhibit desirable behavior. In this paper, we explain how the LP we propose can be thought of as a method for minimizing a version of Bellman error. The important differences here are that our method involves solving a linear – rather than a nonlinear (and nonconvex) – program and that there are performance guarantees that can be made for the outcome.

The next section introduces the problem formulation we will be working with. Section 3 presents the LP approximation algorithm and an error bound. In Section 4, we propose a method for computing a desirable reset distribution. The LP approximation algorithm works with a perturbed version of the MDP. Errors introduced by this perturbation are studied in Section 5. A closing section discusses relations to our prior work on LP approaches to approximate DP [6, 8].

## 2  Problem Formulation and Perturbation Via Restart

Consider an MDP with a countable state space $\mathcal{S}$ and a finite set of actions $\mathcal{A}$ available at each state. Under a control policy $u : \mathcal{S} \mapsto \mathcal{A}$, the system dynamics are defined by a transition probability matrix $P_u \in \Re^{|\mathcal{S}| \times |\mathcal{S}|}$, where for policies $u$ and $\overline{u}$ and states $x$ and $y$, $(P_u)_{xy} = (P_{\overline{u}})_{xy}$ if $u(x) = \overline{u}(x)$. We will assume

that, under each policy $u$, the system has a unique invariant distribution, given by $\pi_u(x) = \lim_{t \to \infty} (P_u^t)_{yx}$, for all $x, y \in \mathcal{S}$.

A cost $g(x, a)$ is associated with each state-action pair $(x, a)$. For shorthand, given any policy $u$, we let $g_u(x) = g(x, u(x))$. We consider the problem of computing a policy that minimizes the average cost $\lambda_u = \pi_u^T g_u$. Let $\lambda^* = \min_u \lambda_u$ and define the *differential value function* $h^*(x) = \min_u \lim_{T \to \infty} E_x^u[\sum_{t=0}^{T}(g_u(x_t) - \lambda^*)]$. Here, the superscript $u$ of the expectation operator denotes the control policy and the subscript $x$ denotes conditioning on $x_0 = x$. It is easy to show that there exists a policy $u$ that simultaneously minimizes the expectation for every $x$. Further, a policy $u^*$ is optimal if and only if $u^*(x) \in \arg\min_u(g(x, a) + \sum_y (P_u)_{xy} h^*(y))$ for all $x \in \mathcal{S}$.

While in principle $h^*$ can be computed exactly by dynamic programming algorithms, this is often infeasible due to the curse of dimensionality. We consider approximating $h^*$ using a linear combination $\sum_{k=1}^{K} r_k \phi_k$ of fixed basis functions $\phi_1, \ldots, \phi_K : \mathcal{S} \mapsto \mathfrak{R}$. In this paper, we propose and analyze an algorithm for computing weights $r \in \mathfrak{R}^K$ to approximate: $h^* \approx \sum_{k=1}^{K} \phi_k(x) r_k$. It is useful to define a matrix $\Phi \in \mathfrak{R}^{|\mathcal{S}| \times K}$ so that our approximation to $h^*$ can be written as $\Phi r$.

The algorithm we will propose operates on a perturbed version of the MDP. The nature of the perturbation is influenced by two parameters: a restart probability $(1 - \alpha) \in [0, 1]$ and a restart distribution $c$ over the state space. We refer to the new system as an $(\alpha, c)$-perturbed MDP. It evolves similarly with the original MDP, except that at each time, the state process restarts with probability $1 - \alpha$; in this event, the next state is sampled randomly according to $c$. Hence, the perturbed MDP has the same state space, action space, and cost function as the original one, but the transition matrix under each policy $u$ are given by $P_{\alpha,u} = \alpha P_u + (1 - \alpha) e c^T$.

We define some notation that will streamline our discussion and analysis of perturbed MDPs. Let $\pi_{\alpha,u}(x) = \lim_{t \to \infty} (P_{\alpha,u}^t)_{yx}$, $\lambda_{\alpha,u} = \pi_{\alpha,u}^T g_u$, $\lambda_\alpha^* = \min_u \lambda_{\alpha,u}$, and let $h_\alpha^*$ be the differential value function for the $(\alpha, c)$-perturbed MDP, and let $u_\alpha^*$ be a policy satisfying $u_\alpha^*(x) \in \arg\min_u(g(x, a) + \sum_y (P_{\alpha,u})_{xy} h_\alpha^*(y))$ for all $x \in \mathcal{S}$. Finally, we will make use of dynamic programming operators $T_{\alpha,u} h = g_u + P_{\alpha,u} h$ and $T_\alpha h = \min_u T_{\alpha,u} h$.

## 3   The New LP

We now propose a new LP that approximates the differential value function of a $(\alpha, c)$-perturbed MDP. This LP takes as input several pieces of problem data:

1. MDP parameters: $g(x, a)$ and $(P_u)_{xy}$ for all $x, y \in \mathcal{S}$, $a \in \mathcal{A}$, $u : \mathcal{S} \mapsto \mathcal{A}$.
2. Perturbation parameters: $\alpha \in [0, 1]$ and $c : \mathcal{S} \mapsto [0, 1]$ with $\sum_x c(x) = 1$.
3. Basis functions: $\Phi = [\phi_1 \cdots \phi_K] \in \mathfrak{R}^{|\mathcal{S}| \times K}$.
4. Slack function and penalty: $\psi : \mathcal{S} \mapsto [1, \infty)$ and $\eta > 0$.

We have defined all these terms except for the slack function and penalty, which we will explain after defining the LP. The LP optimizes decision variables $r \in \mathfrak{R}^K$ and $s_1, s_2 \in \mathfrak{R}$ according to

$$
\begin{aligned}
\text{minimize} \quad & s_1 + \eta s_2 \qquad\qquad\qquad\qquad\qquad (1)\\
\text{subject to} \quad & T_\alpha \Phi r - \Phi r + s_1 \mathbf{1} + s_2 \psi \geq 0 \\
& s_2 \geq 0.
\end{aligned}
$$

It is easy to see that this LP is feasible. Further, if $\eta$ is sufficiently large, the objective is bounded. We assume that this is the case and denote an optimal solution by $(\tilde{r}, \tilde{s}_1, \tilde{s}_2)$. Though the first $|\mathcal{S}|$ constraints are nonlinear, each involves a minimization over actions and therefore can be decomposed into $|\mathcal{A}|$ constraints. This results in a total of $|\mathcal{S}| \times |\mathcal{A}| + 1$ constraints, which is unmanageable if the state space is large. We expect, however, that the solution to this LP can be approximated closely and efficiently through use of constraint sampling techniques along the lines discussed in [7].

We now offer an interpretation of the LP. The constraint $T_\alpha \Phi r - \Phi r - \lambda_\alpha^* \mathbf{1} \geq 0$ is satisfied if and only if $\Phi r = h_\alpha^* + \kappa \mathbf{1}$ for some $\kappa \in \Re$. Terms $(s_1 + \lambda_\alpha^*)\mathbf{1}$ and $s_2 \psi$ can be viewed as *cost shaping*. In particular, they effectively transform the costs $g(x, a)$ to $g(x, a) + s_1 + \lambda_\alpha^* + s_2\psi(x)$, so that the constraint $T_\alpha \Phi r - \Phi r - \lambda_\alpha^* \mathbf{1} \geq 0$ can be met.

The LP can alternatively be viewed as an efficient method for minimizing a form of Bellman error, as we now explain. Suppose that $s_2 = 0$. Then, minimization of $s_1$ corresponds to minimization of $\| \min(T_\alpha \Phi r - \Phi r - \lambda_\alpha^* \mathbf{1}, 0) \|_\infty$, which can be viewed as a measure of (one-sided) Bellman error. Measuring error with respect to the maximum norm is problematic, however, when the state space is large. In the extreme case, when there is an infinite number of states and an unbounded cost function, such errors are typically infinite and therefore do not provide a meaningful objective for optimization. This shortcoming is addressed by the slack term $s_2\psi$. To understand its role, consider constraining $s_1$ to be $-\lambda_\alpha^*$ and minimizing $s_2$. This corresponds to minimization of $\| \min(T_\alpha \Phi r - \Phi r - \lambda_\alpha^* \mathbf{1}, 0) \|_{\infty,1/\psi}$, where the norm is defined by $\|h\|_{\infty,1/\psi} = \max_x |h(x)|/\psi(x)$. This term can be viewed as a measure of Bellman error with respect to a weighted maximum norm, with weights $1/\psi(x)$. One important factor that distinguishes our LP from other approaches to Bellman error minimization [4, 12, 16] is a theoretical performance guarantee, which we now develop.

For any $r$, let $u_{\alpha,r}(x) \in \arg\min_u(g_u(x) + (P_{\alpha,u}\Phi r)(x))$. Let $\pi_{\alpha,r} = \pi_{\alpha,u_{\alpha,r}}$. Let $\lambda_{\alpha,r} = \pi_{\alpha,r}^T g_{u_{\alpha,r}}$. The following theorem establishes that the difference between the average cost $\lambda_{\alpha,\tilde{r}}$ associated with an optimal solution $(\tilde{r}, \tilde{s}_1, \tilde{s}_2)$ to the LP and the optimal average cost $\lambda_\alpha^*$ is proportional to the minimal error that can be attained given the choice of basis functions. A proof of this theorem is provided in the appendix of a version of this paper available at http://www.stanford.edu/ bvr/psfiles/LPnips04.pdf.

**Theorem 3.1.** *If* $\eta \geq (2 - \alpha)\pi_{\alpha,u_\alpha^*}^T \psi$ *then*

$$\lambda_{\alpha,\tilde{r}} - \lambda_\alpha^* \leq \frac{(1 + \beta)\eta \max(\theta, 1)}{1 - \alpha} \min_{r \in \Re^K} \|h_\alpha^* - \Phi r\|_{\infty,1/\psi},$$

*where*

$$\beta = \max_u \|P_{\alpha,u}\|_{\infty,1/\psi} \equiv \max_u \frac{\|P_{\alpha,u}h\|_{\infty,1/\psi}}{\|h\|_{\infty,1/\psi}},$$

$$\theta = \frac{\pi_{\alpha,\tilde{r}}^T(T_\alpha\Phi\tilde{r} - \Phi\tilde{r} + \tilde{s}_1\mathbf{1} + \tilde{s}_2\psi)}{c^T(T_\alpha\Phi\tilde{r} - \Phi\tilde{r} + \tilde{s}_1\mathbf{1} + \tilde{s}_2\psi)}.$$

The bound suggests that the slack function $\psi$ should be chosen so that the basis functions can offer a reasonably sized approximation error $\|h_\alpha^* - \Phi r\|_{\infty,1/\psi}$. At the same time, this choice affects the sizes of $\eta$ and $\beta$. The theorem requires that the penalty $\eta$ be at least $(2 - \alpha)\pi_{\alpha,u_{\alpha*}}^T\psi$. The term $\pi_{\alpha,u_\alpha^*}^T\psi$ is the steady-state

expectation of the slack function under an optimal policy. Note that

$$\beta \leq \max_u \|P_{\alpha,u}\psi\|_{\infty,1/\psi} = \max_{u,x} \frac{(P_{\alpha,u}\psi)(x)}{\psi(x)},$$

which is the maximal factor by which the expectation of $\psi$ can increase over a single time period. When dealing with specific classes of problems it is often possible to select $\psi$ so that the norm $\|h_\alpha^* - \Phi r\|_{\infty,1/\psi}$ as well as the terms $\max_u \|P_{\alpha,u}\|_{\infty,1/\psi}$ and $\pi_{\alpha,u_{\alpha}*}^T \psi$ scale gracefully with the number of states and/or state variables. This issue will be addressed further in a forthcoming full-length version of this paper.

It may sometimes be difficult to verify that any particular value of $\eta$ dominates $(2-\alpha)\pi_{\alpha,u_\alpha*}^T \psi$. One approach to selecting $\eta$ is to perform a line search over possible values of $\eta$, solving an LP in each case, and choosing the value of $\eta$ that results in the best-performing control policy. A simple line search algorithm solves the LP successively for $\eta = 1, 2, 4, 8, \ldots$, until the optimal solution is such that $\tilde{s}_2 = 0$. It is easy to show that the LP is unbounded for all $\eta < 1$, and that there is a finite $\overline{\eta} = \inf\{\eta | \tilde{s}_2 = 0\}$ such that for each $\eta \geq \overline{\eta}$, the solution is identical and $\tilde{s}_2 = 0$. This search process delivers a policy that is at least as good as a policy generated by the LP for some $\eta \in [(2-\alpha)\pi_{\alpha,u^\alpha}^T \psi, 2(2-\alpha)\pi_{\alpha,u^\alpha}^T \psi]$, and the upper bound of Theorem 3.1 would hold with $\eta$ replaced by $2(2-\alpha)\pi_{\alpha,u_\alpha^*}^T \psi$.

We have discussed all but two terms involved in the bound: $\theta$ and $1/(1-\alpha)$. Note that if $c = \pi_{\alpha,\tilde{r}}$, then $\theta = 1$. In the next section, we discuss an approach that aims at choosing $c$ to be close enough to $\pi_{\alpha,\tilde{r}}$ so that $\theta$ is approximately 1. In Section 5, we discuss how the reset probability $1 - \alpha$ should be chosen in order to ensure that policies for the perturbed MDP offer similar performance when applied to the original MDP. This choice determines the magnitude of $1/(1-\alpha)$.

## 4 Fixed Points and Path Following

The coefficient $\theta$ would be equal to 1 if $c$ were equal to $\pi_{\alpha,\tilde{r}}$. We can not to simply choose $c$ to be equal to $\pi_{\alpha,\tilde{r}}$, since $\pi_{\alpha,\tilde{r}}$ depends on $\tilde{r}$, an outcome of the LP which depends on $c$. Rather, arriving at a distribution $c$ such that $c = \pi_{\alpha,\tilde{r}}$ is a fixed point problem. In this section, we explore a path-following algorithm for approximating such a fixed point [9], with the aim of arriving at a value of $\theta$ that is close to one.

Consider solving a sequence indexed by $i = 1, \ldots, M$ of $(\alpha_i, c_i)$-perturbed MDPs. Let $\tilde{r}^i$ denote the weight vector associated with an optimal solution to the LP (1) with perturbation parameters $(\alpha_i, c_i)$. Let $\alpha_1 = 0$ and $\alpha_{i+1} = \alpha_i + \delta$ for $i \geq 1$, where $\delta$ is a small positive step size. For any initial choice of $c_1$, we have $c_1 = \pi_{\alpha_1,\tilde{r}^1}$, since the system resets in every time period. For $i \geq 1$, let $c_{i+1} = \pi_{\alpha_i,\tilde{r}^i}$. One might hope that the change in $c_i$ is gradual, and therefore, $c_i \approx \pi_{\alpha_i,\tilde{r}^i}$ for each $i$.

We can not yet offer rigorous theoretical support for the proposed path following algorithm. However, we will present promising results from a simple computational experiment. This experiment involves a problem with continuous state and action spaces. Though our main result – Theorem 3.1 – applies to problems with countable state spaces and finite action spaces, there is no reason why the LP cannot be applied to broader classes of problems such as the one we now describe. Consider a scalar state process $x_{t+1} = x_t + a_t + w_t$, driven by scalar actions $a_t$ and a sequence $w_t$ i.i.d. zero-mean unit-variance normal random variables. Consider a cost function $g(x,a) = (x-2)^2 + a^2$. We aim at approximating the differential value function using a single basis function $\phi(x) = x^2$. Hence, $(\Phi r)(x) = rx^2$, with $r \in \Re$. We will use a slack function $\psi(x) = 1 + x^2$ and penalty $\eta = 5$. The special structure of this

problem allows for exact solution of the LP (1) as well as the exact computation of the parameter $\theta$, though we will not explain here how this is done. Figure 1 plots $\theta$ versus $\alpha$, as $\alpha$ is increased from 0 to 0.99, with $c$ initially set to a zero-mean normal distribution with variance 4. The three curves represent results from using three different step sizes $\delta \in \{0.01, 0.005, 0.0025\}$. Note that in all cases, $\theta$ is very close to 1. Smaller values of $\delta$ resulted in curves being closer to 1: the lowest curve corresponds to $\delta = 0.01$ and the highest curve corresponds to $\delta = 0.0025$.

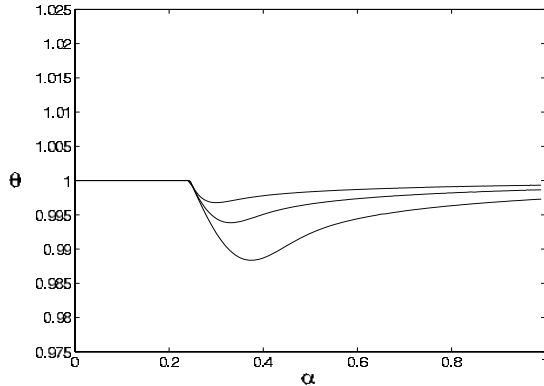

Figure 1: Evolution of $\theta$ with $\delta \in \{0.01, 0.005, 0.0025\}$.

## 5   The Impact of Perturbation

Some simple algebra will show that for any policy $u$,

$$\lambda_{\alpha,u} - \lambda_u = (1 - \alpha) \sum_{t=0}^{\infty} \alpha^t \left( c^T P_u^t g_u - \pi_u^T g_u \right).$$

When the state space is finite $|c^T P_u^t g_u - \pi_u^T g_u|$ decays at a geometric rate. This is also true in many practical contexts involving infinite state spaces. One might think of $m_u = \sum_{t=0}^{\infty} (c^T P_u^t g_u - \pi_u^T g_u)$, as the *mixing time* of the policy $u$ if the initial state is drawn according to the restart distribution $c$. This mixing time is finite if the differences $c^T P_u^t g_u - \pi_u^T g_u$ converge geometrically. Further, we have $|\lambda_{\alpha,u} - \lambda_u| = m_u(1 - \alpha)$, and coming back to the LP, this implies that

$$\lambda_{u_{\alpha,\tilde{r}}} - \lambda_{u^*} \leq \lambda_{\alpha,\tilde{r}} - \lambda_{\alpha,u_{\alpha}^*} + (1 - \alpha)(m_{u_{\alpha,\tilde{r}}} + \max(m_{u^*}, m_{u_{\alpha}^*})).$$

Combined with the bound of Theorem 3.1, this offers a performance bound for the policy $u_{\alpha,\tilde{r}}$ applied to the original MDP. Note that when $c = \pi_{\alpha,\tilde{r}}$, in the spirit discussed in Section 4, we have $m_{u_{\alpha,\tilde{r}}} = 0$. For simplicity, we will assume in the rest of this section that $m_{u_{\alpha,\tilde{r}}} = 0$ and $m_{u^*} \geq m_{u_{\alpha}^*}$, so that

$$\lambda_{u_{\alpha,\tilde{r}}} - \lambda_{u^*} \leq \lambda_{\alpha,\tilde{r}} - \lambda_{\alpha,u_{\alpha}^*} + (1 - \alpha)m_{u^*}.$$

Let us turn to discuss how $\alpha$ should be chosen. This choice must strike a balance between two factors: the coefficient of $1/(1 - \alpha)$ in the bound of Theorem 3.1 and the loss of $(1 - \alpha)m_{u^*}$ associated with the perturbation. One approach is to fix some $\epsilon > 0$ that we are willing to accept as an absolute performance loss, and then choose $\alpha$ so that $(1 - \alpha)m_{u^*} \leq \epsilon$. Then, we would have $1/(1 - \alpha) \geq m_{u^*}/\epsilon$. Note that the term $1/(1 - \alpha)$ multiplying the right-hand-side of the bound can then be thought of as a constant multiple of the mixing time of $u^*$. An important open question is whether it is possible to design an approximate DP algorithm and establish for that algorithm an error bound that does not depend on the mixing time in this way.

## 6 Relation to Prior Work

In closing, it is worth discussing how our new algorithm and results relate to our prior work on LP approaches to approximate DP [6, 8]. If we remove the slack function by setting $s_2$ to zero and let $s_1 = -(1-\alpha)c^T\Phi r$, our LP (1) becomes

$$
\begin{aligned}
\text{maximize} \quad & c^T\Phi r && (2)\\
\text{subject to} \quad & \min_u(g_u + \alpha P_u\Phi r) - \Phi r \geq 0,
\end{aligned}
$$

which is precisely the LP considered in [6] for approximating the optimal cost-to-go function in a discounted MDP with discount factor $\alpha$. Let $\hat{r}$ be an optimal solution to (2). For any function $V : \mathcal{S} \mapsto \Re^+$, let $\beta_V = \alpha\|\max_u P_u V\|_{\infty,1/V}$. We call $V$ a *Lyapunov function* if $\beta_V < 1$. The following result can be established using an analysis entirely analogous to that carried out in [6]:

**Theorem 6.1.** *If $\beta_{\Phi v} < 1$ and $\Phi v' = \mathbf{1}$ for some $v, v' \in \Re^K$. Then,*

$$
\lambda_{\alpha,\hat{r}} - \lambda_\alpha^* \leq \frac{2\theta c^T\Phi v}{1 - \beta_{\Phi v}} \min_{r\in\Re^K} \|h_\alpha^* - \Phi r\|_{\infty,1/\Phi v}.
$$

A comparison of Theorems 3.1 and 6.1 reveals benefits afforded by the slack function. We consider the situation where $\psi = \Phi v$, which makes the bounds directly comparable. An immediate observation is that, even though $\psi$ and $\Phi v$ play analogous roles in the bounds, $\psi$ is not required to be a Lyapunov function. In this sense, Theorem 3.1 is stronger than Theorem 6.1. Moreover, if $\eta = \pi_{\alpha,u_\alpha^*}^T\psi$, we have

$$
\frac{\eta}{1-\alpha} = c^T(I - \alpha P_{u_\alpha^*})^{-1}\psi \leq \max_u c^T(I - \alpha P_u)^{-1}\Phi v \leq \frac{c^T\Phi v}{1 - \beta_V}.
$$

Hence, the first term – which appears in the bound of Theorem 6.1 – grows with the largest mixing time among all policies, whereas the second term – which appears in the bound of Theorem 3.1 – only depends on the mixing time of an optimal policy.

As discussed in [6], appropriate choice of $c$ – there referred to as the *state-relevance weights* – can be important for the error bound of Theorem 6.1 to scale well with the number of states. In [8], it is argued that some form of weighting of states in terms of a metric of relevance should continue to be important when considering average cost problems. An LP-based algorithm is also presented in [8], but the results are far weaker than the ones we have presented in this paper, and we suspect that the that LP-based algorithm of [8] will not scale well to high-dimensional problems.

Some guidance is offered in [6] regarding how $c$ might be chosen. However, this is ultimately left as a manual task. An important contribution of this paper is the path-following algorithm proposed in Section 4, which aims at automating an effective choice of $c$.

## Acknowledgments

This research was supported in part by the NSF under CAREER Grant ECS-9985229 and by the ONR under grant MURI N00014-00-1-0637.

## References

[1] D. Adelman, "A Price-Directed Approach to Stochastic Inventory/Routing," preprint, 2002, to appear in *Operations Research*.

[2] L. C. Baird, "Residual Algorithms: Reinforcement Learning with Function Approximation," ICML, 1995.

[3] D. P. Bertsekas and J. N. Tsitsiklis, *Neuro-Dynamic Programming*, Athena Scientific, Bellmont, MA, 1996.

[4] D. P. Bertsekas, *Dynamic Programming and Optimal Control*, second edition, Athena Scientific, Bellmont, MA, 2001.

[5] J. A. Boyan and A. W. Moore, "Generalization in Reinforcement Learning: Safely Approximating the Value Function," NIPS, 1995.

[6] D. P. de Farias and B. Van Roy, "The Linear Programming Approach to Approximate Dynamic Programming," *Operations Research*, Vol. 51, No. 6, November-December 2003, pp. 850-865. Preliminary version appeared in NIPS, 2001.

[7] D. P. de Farias and B. Van Roy, "On Constraint Sampling in the Linear Programming Approach to Approximate Dynamic Programming," *Mathematics of Operations Research,* Vol. 29, No. 3, 2004, pp. 462–478.

[8] D.P. de Farias and B. Van Roy, "Approximate Linear Programming for Average-Cost Dynamic Programming," NIPS, 2003.

[9] C. B. Garcia and W. I. Zangwill, *Pathways to Solutions, Fixed Points, and Equilibria*, Prentice-Hall, Englewood Cliffs, NJ, 1981.

[10] G. J. Gordon, "Stable Function Approximation in Dynamic Programming," ICML, 1995.

[11] C. Guestrin, D. Koller, R. Parr, and S. Venkataraman, "Efficient Solution Algorithms for Factored MDPs," *Journal of Artificial Intelligence Research*, Volume 19, 2003, pp. 399-468. Preliminary version appeared in NIPS, 2001.

[12] M. E. Harmon, L. C. Baird, and A. H. Klopf, "Advantage Updating Applied to a Differential Game," NIPS 1995.

[13] R. Munos, "Error Bounds for Approximate Policy Iteration," ICML, 2003.

[14] J. N. Tsitsiklis and B. Van Roy, "Feature-Based Methods for Large Scale Dynamic Programming," *Machine Learning*, Vol. 22, 1996, pp. 59-94.

[15] D. Schuurmans and R. Patrascu, "Direct Value Approximation for Factored MDPs," NIPS, 2001.

[16] P. J. Schweitzer and A. Seidman, "Generalized Polynomial Approximation in Markovian Decision Processes," *Journal of Mathematical Analysis and Applications*, Vol. 110, '985, pp. 568-582.
